# Certified Adversarial Robustness via Randomized $\alpha$-Smoothing for Regression Models

**Aref Miri Rekavandi**
University of Melbourne
aref.mirirekavandi@unimelb.edu.au

**Farhad Farokhi**
University of Melbourne
farhad.farokhi@unimelb.edu.au

**Olga Ohrimenko**
University of Melbourne
oohrimenko@unimelb.edu.au

**Benjamin I.P. Rubinstein**
University of Melbourne
benjamin.rubinstein@unimelb.edu.au

## Abstract

Certified adversarial robustness of large-scale deep networks has progressed substantially after the introduction of randomized smoothing. Deep net classifiers are now provably robust in their predictions against a large class of threat models, including $\ell_1$, $\ell_2$, and $\ell_\infty$ norm-bounded attacks. Certified robustness analysis by randomized smoothing has not been performed for deep regression networks where the output variable is continuous and unbounded. In this paper, we extend the existing results for randomized smoothing into regression models using powerful tools from robust statistics, in particular, $\alpha$-trimming filter as the smoothing function. Adjusting the hyperparameter $\alpha$ achieves a smooth trade-off between desired certified robustness and utility. For the first time, we propose a benchmark for certified robust regression in visual positioning systems using the *Cambridge Landmarks* dataset where robustness analysis is essential for autonomous navigation of AI agents and self-driving cars. Code is publicly available at https://github.com/arekavandi/Certified_adv_RRegression/.

## 1 Introduction

Adversarial examples first swayed the narrative on deep models over a decade ago [2, 26]. Where deep nets had demonstrated remarkable generalization on classically challenging tasks [10], these small perturbations to an input sample that make no apparent change to the input's semantics or true class, yield high rates of misclassifications. While defenses had so far not tipped the balance away from the attacker, the combination of Certified Robustness (CR) with adversarial training has excited the security and ML communities recently [15]. For a given model and input sample, CR can guarantee the absence of any adversarial examples in close vicinity of the sample. Randomized Smoothing (RS) is a widely used technique for CR as it scales to arbitrary large-scale models as it requires only black-box access to model evaluations [12, 6]. Despite attacks targeting ML tasks beyond classification [5], little is known of certification (or RS) for other standard ML tasks such as regression. Robustness analysis for regression has been examined through Lipschitz continuity [27] which is only feasible for small-scale regression models with full access to the models' parameters and certain types of activation functions.

In this paper, we present a framework for black-box certification of arbitrary regression models with either bounded or unbounded outputs. Our results extend the findings of [17] which considers the class of bounded-output regression models with large sample sizes in the evaluation stage. While they introduce averaging as an aggregation operator in randomized smoothing for regression, without limits on output range, averaged predictions may not be stable. This boundedness assumption, however,

limits the applicability of their certificates. We present a superior approach through smoothing by $\alpha$-trimming filter. Complementing experiments with synthetic data, we benchmark our CR and RS approaches with the *Cambridge Landmarks* [9] dataset and DSAC* framework [3] for camera re-localization. The use of these benchmarks may be of independent interest for CR research.

## 2 Preliminaries

A base regression model parameterized with $\boldsymbol{\theta}$ is denoted by $\mathbf{f}_\theta : \mathbb{R}^d \to \mathbb{R}^t$, where $d$ and $t$ are the input and output dimensions: $\mathbf{f}_\theta(\mathbf{x})$ maps input $\mathbf{x}$ to multivariate output $\mathbf{y}$. We will use $\mathbf{g}_\alpha(\mathbf{x})$ to denote the smoothed version of $\mathbf{f}_\theta(\mathbf{x})$, as defined later. The neighborhood centered around point $\mathbf{z} \in \mathbb{R}^s$ with radius $\epsilon$ with respect to a given dissimilarity function is denoted by $\mathbf{N}(\mathbf{z}, \epsilon)$, where the dissimilarity function can be e.g., $\ell_p$ norms as well as functional divergences such as KL or Bregman when dealing with normalized $\mathbf{z}$. In other words, for any $\mathbf{z} \in \mathbb{R}^s$,

$$\mathbf{N}(\mathbf{z}, \epsilon) = \{\mathbf{z}' \in \mathbb{R}^s \mid \mathrm{diss}(\mathbf{z}, \mathbf{z}') \le \epsilon\}, \tag{1}$$

where $\mathrm{diss}(.,.)$ in general, can be any metric or function that the user is interested, and $\mathrm{diss}_x$ (or $\mathrm{diss}_y$) indicates dissimilarity in the input (or output) space. Throughout this paper, neighborhoods in input space are defined for all dimensions simultaneously as in Eq. (1), neighborhoods for outputs are analyzed separately using the neighboring function $\mathbf{N}_\mathbf{y}(\mathbf{y}, \epsilon_y) = \prod_{i=1}^{t} \mathbf{N}_y(\mathbf{y}_i, \epsilon_{y_i})$. The $\ell_p$-norm ($p \ge 1$) of a vector is denoted and defined as $\|\mathbf{x}\|_p = (\sum_i |\mathbf{x}_i|^p)^{1/p}$ where $\mathbf{x}_i$ indicates the $i^{th}$ entry of $\mathbf{x}$. We denote the multivariate normal distribution with mean $\mathbf{m}$ and covariance $\sigma^2\mathbf{I}$ as $\mathcal{N}(\mathbf{m}, \sigma^2\mathbf{I})$, where $\mathbf{I}$ is the identity matrix. We denote the standard Gaussian CDF by $\Phi(\cdot)$. $\mathbb{P}\{\cdot\}$ and $\mathbb{E}\{\cdot\}$ denote the probability and expected value operators, respectively. Finally, $[\![t]\!]$ indicates the set $\{1, 2, \cdots, t\}$ and $[\cdot]$ rounds to the closest integer value.

**Threat model.** We consider a defender with only black-box access to model evaluations at test time: given an input point they may observe the output regressed value, but not model structure, gradients, parameters, or learning hyperparameters. On the other hand, we consider adversaries that have full information access to the model and certification algorithm. However, the attacker is limited to perturbing input data within small neighborhoods. This matches the typical threat model found in previous works on randomized smoothing in classification [6].

**Randomized smoothing.** Randomized smoothing is based on the ensemble of model outputs obtained over different perturbed inputs and is among the few techniques that are scalable to arbitrary, large models. Randomized smoothing was first adopted in seminal works of [6, 14, 12] for classification tasks to derive the maximum radii of input perturbations which maintain an invariant output prediction. In particular, in [6] it was shown that for the base classifier $\mathbf{f}_\theta(\mathbf{x})$, the new smoothed classifier $g(\mathbf{x}) = \arg\max_{c \in \mathbf{Y}} \mathbb{P}(f_\theta(\mathbf{x} + \mathbf{e}) = c)$, $\mathbf{e} \sim \mathcal{N}(\mathbf{0}, \sigma^2\mathbf{I})$, is certifiably robust against any $\ell_2$-norm-bounded adversary with radius $\epsilon = \frac{\sigma}{2}(\Phi^{-1}(\underline{p_A}) - \Phi^{-1}(\overline{p_B}))$ where $\underline{p_A} \ge \overline{p_B}$ are any lower bound of majority class scores, and any upper bound of runner-up class scores, respectively. While randomized smoothing has mostly been explored for classification problems with categorical outputs [8, 30, 11, 31, 28, 13], more recently research has begun to consider regression [17, 20, 7, 4, 16]. These works all exhibit some limitation, including: universality, theoretical support, practical applicability, reliance on large sample sizes of data points, bounded outputs, or analysis through the lens of certifying classification. Further explanation on the differences among these studies can be found in Section 5.

## 3 Method

### 3.1 Certified Regression

For many years, robustness analysis of regression models or estimators has been investigated using tools, such as Lipschitz continuity [27] or influence functions [23], where full access to the model parameters and its derivatives were required. Due to a lack of a proper definition of robustness in black-box access setup for large models, in contrast to classification problems, certified robustness has not been fully developed for regression problems. Until recently, motivated by probabilistic certified classification [18], the robustness definition for regression models has been introduced in [17], and some input perturbation bounds for base regression models, as well as their smoothed version with bounded outputs assumption using sample averaging, have been derived. In this paper, we extend

these results for unbounded outputs in a small sample regime using a much more general form of smoothing function. We use the same definition of robustness in regression tasks given by:

**Definition 1.** *(Probabilistic Robustness Certificate) [17]. Given an example $(\boldsymbol{x}, \boldsymbol{y})$, a (possibly) randomized regression function $\boldsymbol{g}(\boldsymbol{x}) : \mathbb{R}^d \to \mathbb{R}^t$ is said to be certifiably robust with probability $0 \le P \le 1$ in the randomness of $\boldsymbol{g}$, with respect to the given input and output dissimilarity functions with radii $\epsilon_x, \epsilon_{y_1}, \cdots, \epsilon_{y_t}$, If $\forall \boldsymbol{x}' \in \mathbf{N}_x(\boldsymbol{x}, \epsilon_x)$*

$$\min_{i \in [\![t]\!]} \mathbb{P}\big\{ diss_y(\boldsymbol{g}(\boldsymbol{x}')_i, \boldsymbol{y}_i) \le \epsilon_{y_i} \big\} \ge P. \tag{2}$$

This definition of robustness in black-box functions requires first analyzing the robustness of the base regressor and then establishing theoretical results for smoothing functions applied to this base regression as a wrapper. Based on Definition 1, users can define a region for $i^{th}$ continuous output variable by $\mathbf{N}_y(\mathbf{y}_i, \epsilon_{y_i})$ or in other words $\{z \mid diss_y(z, \mathbf{y}_i) \le \epsilon_{y_i}\}$ as the accepted/valid region where the output prediction can fit in without being considered as a wrong prediction. The term accepted/valid region will be used in the rest of the paper to refer to the output neighborhood. This region is set by the user around $f(\mathbf{x})$ to determine how much deviation is acceptable. For example, in camera- re-localization in a 3D scene with size $100m \times 100m$, the user might reasonably accept up to 0.5m deviations in the predictions. The following result has been provided for the base regression function:

**Theorem 1.** *(Certification of General Models Against $\ell_2$-Bounded Attack) [17]. Let $\boldsymbol{f}_\theta(\boldsymbol{x}) : \mathbb{R}^d \to \mathbb{R}^t$ be a (possibly) randomized base regressor and $\boldsymbol{e} \sim \mathcal{N}(\boldsymbol{0}, \sigma^2 \boldsymbol{I})$. Suppose for some example $(\boldsymbol{x}, \boldsymbol{y})$,*

$$\mathbb{P}\{diss_y(\boldsymbol{f}_\theta(\boldsymbol{x} + \boldsymbol{e})_i, \boldsymbol{y}_i) \le \epsilon_{y_i}\} \ge \underline{p_{A_i}}, \forall i \in [\![t]\!] \tag{3}$$

*where $\underline{p_{A_i}}$ is the lower bound on the probability of accepting prediction in the $i^{th}$ output variable. Then referring to definition Eq. (2), $\boldsymbol{f}_\theta(\boldsymbol{x} + \boldsymbol{e})$ is probabilistic certifiably robust at example $(\boldsymbol{x}, \boldsymbol{y})$, for a $\ell_2$-norm dissimilarity in the input, chosen probability $P \le \min_{i \in [\![t]\!]} \underline{p_{A_i}}$, output radii $\epsilon_{y_1}, \ldots, \epsilon_{y_t}$ and input radius*

$$\epsilon_x = \min_{i \in [\![t]\!]} \sigma \big( \Phi^{-1}(\underline{p_{A_i}}) - \Phi^{-1}(P) \big). \tag{4}$$

The above result indicates a strong similarity between the certification of regression models and classification task since both radii are proportional to $\sigma$ and $\Phi^{-1}(p_A)$. It is worth investigating the equivalence of these two tasks in terms of robustness radius formulation, i.e., Eq. (4):

**Corollary 1.** *(On the Equivalence of Regression and Classification) The certificate radius for a univariate regression model, i.e., $\boldsymbol{f}_\theta(\boldsymbol{x}) : \mathbb{R}^d \to \mathbb{R}$ using the robustness definition (2) with a user-chosen $P = 50\%$, is the same as the certified radius for a smoothed classifier made of $\boldsymbol{f}_\theta(\boldsymbol{x}) : \mathbb{R}^d \to \mathbb{R}$ as the base binary classifier and dividing the space into class A and class not A, where A denotes the case where the output is within the defined output neighborhood.*

*Proof.* Starting from (4) as the regression certificate bound and setting $P = 50\%$, the regression certificate radius becomes

$$\epsilon_x = \sigma \Phi^{-1}(\underline{p_A}), \tag{5}$$

which is exactly the radius derived for a binary classifier using the same smoothed function with the majority voting as stated in [6]. $\qquad\square$

This is an intuitive result stating that robustness in regression tasks can be reduced to the classification task when $P$ is set to $50\%$, where the output of a regression model is in the maximum uncertainty level. In this case, there is a tie between class $A$ and not $A$ and this is exactly where the output of a classification model might change and break the robustness definition of the given classifier. Now we extend the results in Theorem 1 to the $\ell_p$ attack with proof in Appendix A.

**Proposition 1.** *(Certification of $\boldsymbol{f}_\theta(\boldsymbol{x})$ Against $\ell_p$ Attack). Let $\boldsymbol{f}_\theta(\boldsymbol{x}) : \mathbb{R}^d \to \mathbb{R}^t$ be a (possibly) randomized base regressor and $\boldsymbol{e} \sim \mathcal{N}(\boldsymbol{0}, \sigma^2 \boldsymbol{I})$. Suppose*

$$\mathbb{P}\{diss_y(\boldsymbol{f}_\theta(\boldsymbol{x} + \boldsymbol{e})_i, \boldsymbol{y}_i) \le \epsilon_{y_i}\} \ge \underline{p_{A_i}}, \forall i \in [\![t]\!], \tag{6}$$

*where $\underline{p_{A_i}}$ is the lower bound on the probability of accepting prediction in $i^{th}$ output variable. Then using (2) as the definition of certified robustness, $\boldsymbol{f}_\theta(\boldsymbol{x} + \boldsymbol{\delta} + \boldsymbol{e})$, $\forall \|\boldsymbol{\delta}\|_p \le \epsilon_x$ $(p \ge 2)$ is within the accepted region, with the user-defined probability $P \le \underline{p_{A_i}}, \forall i \in [\![t]\!]$, where*

$$\epsilon_x = \min_{i \in [\![t]\!]} \frac{\sigma}{d^{\frac{1}{2} - \frac{1}{p}}} \big( \Phi^{-1}(\underline{p_{A_i}}) - \Phi^{-1}(P) \big). \tag{7}$$

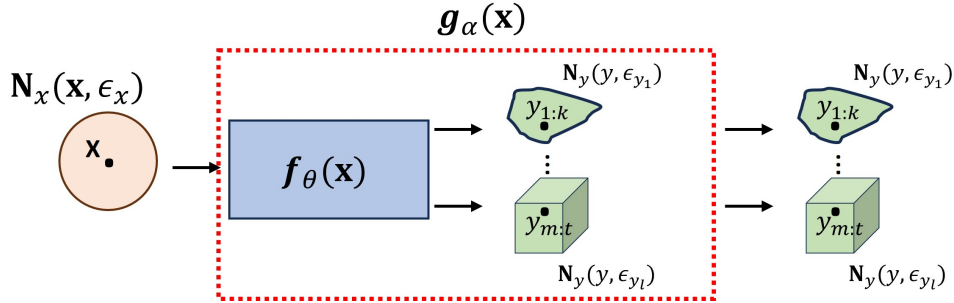

Figure 1: General schematic of how $\alpha$-trimming can be applied to the base regressor with $\ell_2$-norm ball (can be any $\ell_p$ norm in this paper and can be any neighboring function in general) defined for input vicinity and any form of convex (in this paper) or nonconvex (in general) set for the output vicinity. Furthermore, outputs can be examined separately (in this paper) or jointly with other outputs (in the general case as denoted by $\mathbf{y}_{1:k}$ and $\mathbf{y}_{m:t}$).

It is worth pointing out that the upper bound of input perturbation in (7) is obtained by Gaussian smoothing, meaning that we only evaluate the model once with Gaussian smoothing, but the guarantee is valid for any $\ell_p$ attack ($p \geq 2$).

## 3.2 Smoothing in Regression

Definition of the smoothing function in the classification task, i.e., $g(\mathbf{x}) = \arg\max_{c \in \mathbf{Y}} \mathbb{P}(f(\mathbf{x} + \mathbf{e}) = c)$, $\mathbf{e} \sim \mathcal{N}(\mathbf{0}, \sigma^2 \mathbf{I})$, is not feasible for application to regression and requires adjustments. One immediate change can be the use of an average function to compute $\mathbf{g}(\mathbf{x})$, i.e., $\mathbf{g}(\mathbf{x}) = \mathbb{E}\{\mathbf{f}_\theta(\mathbf{x} + \mathbf{e})\}$, $\mathbf{e} \sim \mathcal{N}(\mathbf{0}, \sigma^2 \mathbf{I})$ or its Monte Carlo estimation. However, as shown in [17], even a single adversarial point (in unbounded scenarios) can entirely shift the result of averaging into the invalid zone (from the user's perspective). This behavior is known as the zero breakdown point of averaging in robust statistics [32]. To deal with such a worst-case scenario, in [17] the outputs of regression models were assumed to be bounded. Although this assumption helped to derive certificate bounds around the input, it was shown for some cases where these considered bounds in the output are loose, the certificate bound in the input becomes worse than the base regression model. This motivates use of a better smoothing function that can also tolerate unbounded outputs. Therefore, we use the $\alpha$-trimming filter [1] to estimate the continuous output variable. Suppose that $F$ is a finite set of N numbers (sorted in ascending order). The $\alpha$-trimmed mean of F is obtained by removing $\alpha$ fraction of the samples ($0 \leq \alpha < 0.5$) from the high and low ends of the sorted set, and computing the average of the remaining values. In the extreme cases, when $\alpha \to \frac{1}{2}$, $\alpha$-trimming is equivalent to median filtering [16], and when $\alpha = 0$ it reduces to classical averaging [17]. In other words, we use the following *general* form of smoothing function denoted by $\mathbf{g}_\alpha(\mathbf{x})$:

$$\mathbf{g}_\alpha(\mathbf{x})_i = \frac{1}{n - 2[\alpha n]} \sum_{k=[\alpha n]+1}^{n-[\alpha n]} \mathbf{F}_\theta^k(\mathbf{x} + \mathbf{e})_i, \forall i \in [\![t]\!] \text{ where } \mathbf{e} \sim \mathcal{N}(\mathbf{0}, \sigma^2 \mathbf{I}) \tag{8}$$

where $n$ is the sampling size, $\mathbf{F}_\theta(\mathbf{x} + \mathbf{e})_i$ is the sorted form of $\mathbf{f}_\theta(\mathbf{x} + \mathbf{e})_i$, and $k$ is the index of the order statistics. In the above, we draw $n$ realization from $\mathbf{e}$ to construct the set values. Here $\alpha$ is a hyperparameter and can be tuned based on the user-chosen value $P$ or level of smoothing. One of the primary uses of the $\alpha$-trimming filter is in data preprocessing and outlier rejection. The adjustment of $\alpha$ in this context typically relies on prior knowledge about the proportion of data points that deviate from the nominal distribution [22]. While this prior knowledge may not always be accurate, it has been widely utilized to reduce the sensitivity of estimators. Another method for tuning $\alpha$ in parameter estimation is to consider efficiency at the nominal density. For instance, if no outliers are present in the dataset, one may set $\alpha$ to achieve an estimation that closely matches the performance of its maximum likelihood counterpart [24].

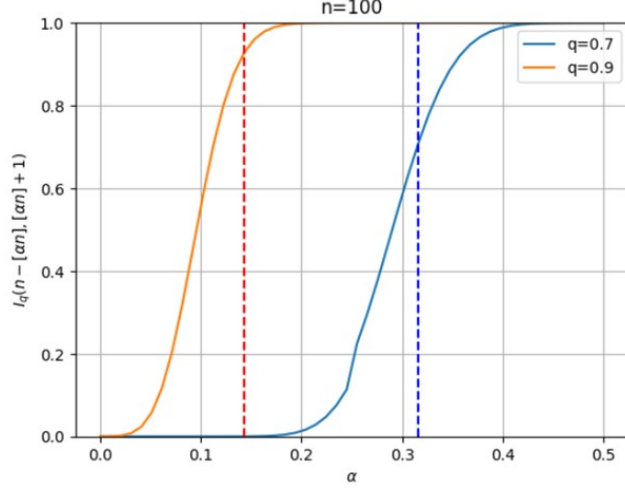

Figure 2: Improvement in the probability of observing predictions within defined accepted region using $\alpha$-trimming for different $\alpha$ values.

### 3.3 Certified Regression Against $\ell_p$ Attack

In this section, we use $\alpha$-trim smoothing function $\mathbf{g}_\alpha(\mathbf{x})$ as a wrapper around the base regression model using the same definition of the vicinity sets in both input and output as shown in Figure 1. Consequently, everything within the red dashed box is related to base regression certification. The corresponding output regions outside the red dashed box represent the certification analysis for the smoothed function with the accepted regions remaining consistent with those in the base regression model. Note that for the following results (proof in Appendix B), the neighborhood in the output can be in any convex region, but the input perturbation is only considered to be within a $\ell_p$ ball around $\mathbf{x}$.

**Theorem 2.** *(Certification of $\mathbf{g}_\alpha(\mathbf{x})$ for fixed range $\ell_p$ Attack). Let $\mathbf{f}_\theta(\mathbf{x}) : \mathbb{R}^d \to \mathbb{R}^t$ be a (possibly) randomized base regressor, and let $\mathbf{g}_\alpha(\mathbf{x})$ be the one defined in (8) with $0 \leq \alpha \leq 1/2$. Let us assume the accepted region (set) for each output target variable is convex and the following inequality holds for $\mathbf{e} \sim \mathcal{N}(\mathbf{0}, \sigma^2 \mathbf{I})$, a user-defined $\boldsymbol{\epsilon}_y$ and $\forall \|\boldsymbol{\delta}\|_p \leq \epsilon_x$ $(p \geq 2)$ and $\forall i \in [\![t]\!]$:*

$$\mathbb{P}\{diss_y(\mathbf{f}_\theta(\mathbf{x} + \boldsymbol{\delta} + \mathbf{e})_i, \mathbf{y}_i) \leq \epsilon_{y_i}\} \geq q, \tag{9}$$

*where $0 \leq q \leq 1$, then $\forall n > 0$ we have*

$$\mathbb{P}\{diss_y(\mathbf{g}_\alpha(\mathbf{x} + \boldsymbol{\delta})_i, \mathbf{y}_i) \leq \epsilon_{y_i}\} \geq I_q(n - [\alpha n], [\alpha n] + 1), \forall i \in [\![t]\!] \tag{10}$$

*where $I_q(a, b)$ is the regularized incomplete beta function defined as $I_q(a, b) = \frac{1}{B(a,b)} \int_0^q t^{a-1}(1 - t)^{b-1} dt$ and $B(a, b)$ is the complete beta function.*

Theorem 2 states that after applying the $\alpha$-trimming filter, the probability that the average prediction of the regression network satisfies the user-chosen constraint, in the worst case scenario changes from $q$ to $I_q(n - [\alpha n], [\alpha n] + 1)$.

The result in Theorem 2 is general and several intuitive and interesting cases are worth mentioning, (i) in the case where $q = 0$, regardless of the number of samples and the value of trimming parameter $\alpha$, $I_0(n - [\alpha n], [\alpha n] + 1) = 0$, meaning that when all the generated outputs by the base regression model are outside the accepted region, there is no chance that $\alpha$-trimming can generate a valid result. (ii) When $q \to 1$, $I_q(n - [\alpha n], [\alpha n] + 1) \to 1$, meaning that when almost all of the generated outputs are valid, the $\alpha$-trimming approach with probability 1 returns a valid result regardless of the values of $\alpha$ and $n$. (iii) When $\alpha \to 0$, $I_q(n - [\alpha n], [\alpha n] + 1) \to q^n$, meaning that if no trimming is applied, one single large invalid output can make the result invalid, then to get valid output, all the generated outputs by base regression model should be within the defined acceptable zone, where the chance of this event is $q^n$. Note that $q^n$ for $\alpha = 0$ is much lower than the result obtained in the case of bounded output for the same smoothing in [17] which indicates the strong impact of the output range on the provided certificate.

**Algorithm 1: Pseudocode** for prediction and certification of smoothed regression model $\mathbf{g}_\alpha$ at $\mathbf{x}$.

**Input** : $\mathbf{x}$, $p$ in $\ell_p$ norm, $\sigma$, $P$, $n$, $\alpha$, $\beta$, $\epsilon_y$, $\mathbf{f}_\theta(.)$
**Output** : Continuous variably $\hat{\mathbf{y}}$

1 – Draw $n$ noise samples using $\mathbf{e} \sim \mathcal{N}(\mathbf{0}, \sigma^2\mathbf{I})$ and pass the noise-corrupted version of $\mathbf{x}$ through $\mathbf{f}_\theta(.)$, and save the output values.
2 – $n_{A_i} \leftarrow$ number of accepted outputs $\forall i \in [\![t]\!]$.
3 – Estimate $\underline{p_{A_i}}$ for $\forall i \in [\![t]\!]$ using Clopper-Pearson interval prediction as stated in [17] with confidence $1 - \frac{\beta}{2}$.
4 – $\hat{\mathbf{y}}_i \leftarrow$ Apply the $\alpha$-trimming filter (8) to the sorted output values $\forall i \in [\![t]\!]$.
5 – $\hat{\mathbf{y}}_i$ is valid with probability $I_{p_{A_i}}(n - [\alpha n], [\alpha n] + 1)$ for $\forall i \in [\![t]\!]$.
6 – The certification radius of this prediction is $\epsilon_x = \min_{i \in [\![t]\!]} \frac{\sigma}{d^{\frac{1}{2} - \frac{1}{p}}} \left( \Phi^{-1}(\underline{p_{A_i}}) - \Phi^{-1}(I_{n,\alpha}^{-1}(P)) \right)$.

For other values of $\alpha$, it is important to note that enhancing the robustness of the base regression model is not always guaranteed unless a certain minimum amount of trimming is performed. In the following proposition, we investigate the certificate improvement of $\alpha$-trimming (with conservative $\alpha$) over the base regression model with proof in Appendix C.

**Proposition 2.** *(Robustness Improvement via $\alpha$-trimming). For any $n \geq 1$, $0 \leq q \leq 1$, if there exist an $\alpha$ such that $\alpha^+ \leq \alpha < 1/2$, where*

$$\alpha^+ := \frac{I_q^{-1}(q) - 1/2}{n}, \tag{11}$$

*and where $I_q^{-1}(q)$ is the inverse of $I_q(n - x, x + 1)$ with respect to $x$, then the following inequality holds:*

$$I_q(n - [\alpha n], [\alpha n] + 1) \geq q. \tag{12}$$

Proposition 2 states that for sufficiently large $\alpha$ values, the $\alpha$-trimming averaging technique, improves the probability of robustness and for larger values of $\alpha$, the result gets better and better. As an example, suppose for a deep network that $90\%$ of the generated outputs are valid, i.e., $q = 0.9$. Assuming drawing only $50$ samples to obtain the certificate, then $I_{0.9}^{-1}(0.9) \approx 7.3$ and for any $\alpha$ which meets $\alpha \geq 0.136$, the robustness probability, $I_{0.9}(50 - [50\alpha], [50\alpha] + 1)$ is greater than $q = 0.9$. For instance, when $\alpha = 0.15$ and $\alpha = 0.2$, the robustness probability is $0.94$ and $0.99$, respectively. Intuitively, the obtained range of $\alpha$ shows that to get improvement, $\alpha$ should be at least slightly greater than $1 - q$ ($0.1$ in our example) to be more confident that there will be less chance of observing some samples out of the accepted region defined by the user. Figure 2 shows two different models one with $q = 0.7$ and one with $q = 0.9$. After applying $\alpha$-trimming the obtained probability of validity in the results are shown in blue and orange colours, respectively. In both settings, $\alpha^+$ values are demonstrated in vertical dashed lines and it can be observed that the success rate of the prediction ($I_q(n - [\alpha n], [\alpha n] + 1)$) is always greater than the assumed $q$ values for $\alpha \geq \alpha^+$. As described above, the corresponding $\alpha^+$ values are slightly greater than $1 - q$ to ensure improvement even in worst-case scenarios. Now we must use the above results to propose the bound on $\ell_p$ norm ($p \geq 2$) of the input perturbation for a user-given probability of success (P) when $\alpha$-trimming is in place (proof in Appendix D).

**Theorem 3.** *(Certification of $\mathbf{g}_\alpha(\mathbf{x})$ against $\ell_p$ Attack). Let $\mathbf{f}_\theta(\mathbf{x}) : \mathbb{R}^d \to \mathbb{R}^t$ be a deterministic or random base regressor and let $n \geq 1$, $0 \leq \alpha < 1/2$, $\mathbf{e} \sim \mathcal{N}(\mathbf{0}, \sigma^2\mathbf{I})$ and suppose the $\alpha$-trimming function $\mathbf{g}_\alpha(\mathbf{x})$ defined in (8) is used for smoothing. Then given*

$$\mathbb{P}\{diss_y(\mathbf{f}_\theta(\mathbf{x} + \mathbf{e})_i, \mathbf{y}_i) \leq \epsilon_{y_i}\} \geq \underline{p_{A_i}}, \forall i \in [\![t]\!] \tag{13}$$

*where $\underline{p_{A_i}}$ is the lower bound on the probability of accepting prediction in the $i^{th}$ output variable, then $\mathbf{g}_\alpha(\mathbf{x} + \boldsymbol{\delta})$, $\forall \|\boldsymbol{\delta}\|_p \leq \epsilon_x$ ($p \geq 2$) is within accepted region, i.e., $\mathbf{N_y}(\mathbf{y}, \epsilon_y) = \prod_{i=1}^{t} \mathbf{N}_y(\mathbf{y}_i, \epsilon_{y_i})$, with the user-defined probability P, s.t. $I_{n,\alpha}^{-1}(P) \leq \underline{p_{A_i}}, \forall i \in [\![t]\!]$, where*

$$\epsilon_x = \min_{i \in [\![t]\!]} \frac{\sigma}{d^{\frac{1}{2} - \frac{1}{p}}} \left( \Phi^{-1}(\underline{p_{A_i}}) - \Phi^{-1}(I_{n,\alpha}^{-1}(P)) \right), \tag{14}$$

*and where $I_{n,\alpha}^{-1}(x)$ is the inverse of the regularized beta function w.r.t Bernoulli success rate parameter.*

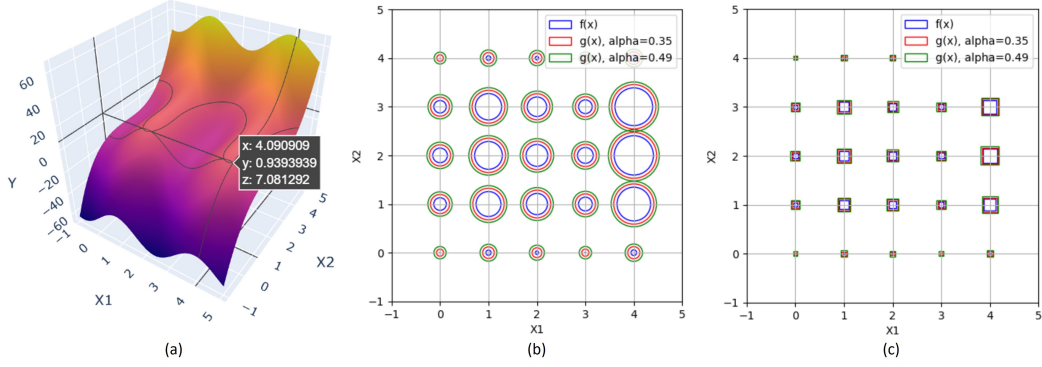

Figure 3: Adopted regression function (a) with the estimated certified radii (against $\ell_2$ and $\ell_\infty$ attacks) for evaluated points in the center for both base and smoothed outputs (b & c).

The above result is valid for any regression model, either with bounded or unbounded outputs, and for large or small $n$. Note that the performance of the base model always impacts the overall performance of the smoothed function, irrespective of whether the setting is classification or regression. In classification tasks, this strong relationship is reflected in the gap between $p_A$ and $\overline{p_B}$. If a base classifier performs poorly under input perturbations, this gap will shrink, diminishing the effectiveness of the final certification. In classification, there is only one other parameter ($\sigma$) that can potentially compensate for this gap. In regression settings, a similar relationship applies: better performance in the base regression model results in a better value for $q$. However, as demonstrated in Theorem 3, the final performance is also influenced by $\sigma$, $n$ and importantly $\alpha$. Using an appropriate value for $\alpha$ can significantly enhance the certification even when the base regression model performs poorly. The step-by-step process of the method in prediction and certification is depicted in Algorithm 1.

## 4   Experiments

This section provides some numerical results to validate the proposed theorems as well as to show their effectiveness in some real-world problems. All simulations and experiments were conducted using an Intel(R) Core(TM) i7-9750H CPU running at 2.60GHz (with a base clock speed of 2.59GHz) and 16GB of RAM.

**Synthetic Simulations.**   For the first part of the experiments, we utilized the function $f(\mathbf{x}) = 10\sin(2x_1) + 2(x_2 - 2)^3$. Figure 3 (left) illustrates this function for the interval $-1 < x_1, x_2 < 5$. As certification is performed for each point individually, we derive the certified radii for this function at all the integer points in the considered intervals utilizing the formulas (4) and (14) with $P = 0.8$, $\sigma = 0.15$, $\epsilon_y = 6$ using $\ell_1$ norm, $n = 10,000$ at two different rates of $\alpha = 0.35$ and $\alpha = 0.49$. Figure 3 (middle & right) illustrates these radii in a 2D grid for the base regression model as well as its smoothed versions against both $\ell_2$ and $\ell_\infty$ norm attacks. As it can be observed, as expected, for smoother areas these radii become larger, and an increase in $\alpha$ results in an increase in estimated certified radii as clear in Eq. (14). In addition to that, as expected, the certified radii become smaller as $p$ increases in $\ell_p$ norm attacks. Now to examine the derived radii, for each point (25 points in total which are raster ordered) we randomly select an adversarial example within the defined radius (we repeat this process) and then find the empirical probability of obtaining valid outputs using the base model and its smoothed version utilizing $\alpha$-trimming filter ($\alpha = 0.35$) with the same hyperparameters with only 5 samples fed into the $\alpha$-trimming filter. Figure 4 depicts the obtained empirical probabilities for both smoothed and base models in comparison to the required probability of valid outputs defined by the user ($P = 0.8$). As shown, although the base regression model shows large variations in the empirical probabilities, with attacks in smaller neighborhoods, the $\alpha$-trimming filter uniformly improves the empirical probability across all the evaluated points (almost equal to 1) for both types of attacks, despite adversarial examples being drawn from larger neighborhoods.

**Camera Re-localization Task.**   In this application, we evaluate the robustness of the state-of-the-art image-based camera re-localization technique "DSAC*" [3] against adversarial examples. In the

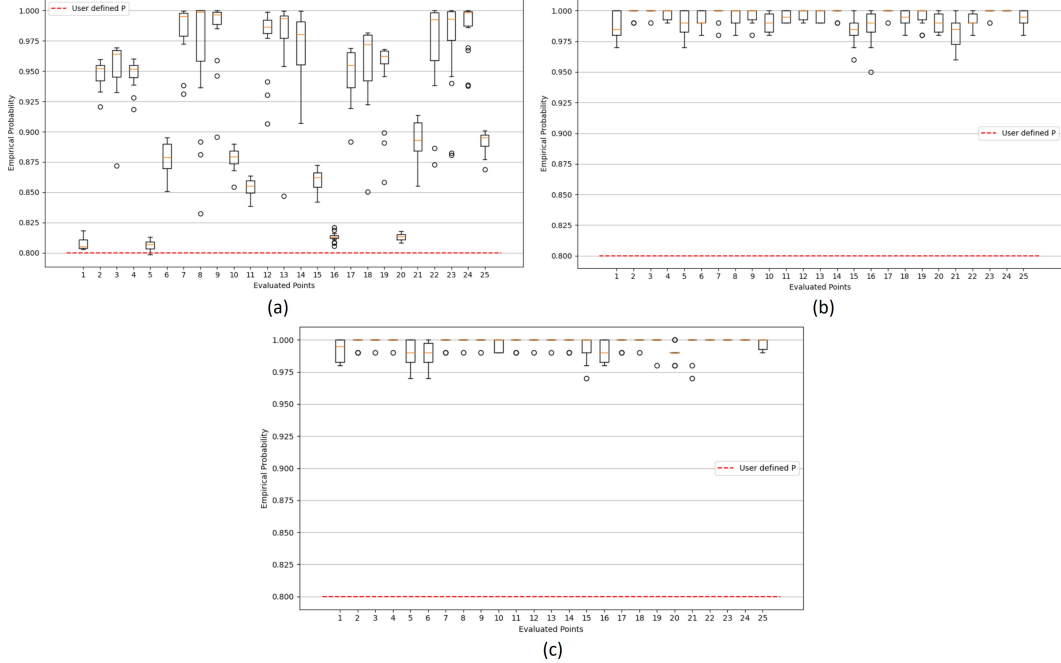

(a)

(b)

(c)

Figure 4: Empirical probability of obtaining valid outputs in comparison with desired probability defined by the user ($P = 0.8$) for the base model against $\ell_2$ attack (a), $g_\alpha(\mathbf{x})$ with $\alpha = 0.35$ against $\ell_2$ attack (b), and $g_\alpha(\mathbf{x})$ with $\alpha = 0.35$ against $\ell_\infty$ attack (c).

camera re-localization pipeline, RGB images are fed into a trained system and the coordinates of the location where the camera was placed to take the images are predicted [21, 19]. For certifying such regression models, we use *Cambridge Landmarks* dataset [9] and in particular 3 of the largest scenes in this popular dataset namely Great Court, King's College, and St. Mary Church. For computing certified error for any image, we used the same formulation as in [17], given by

$$e_K \quad = \quad \|\mathbf{g}_\alpha(\mathbf{x} + \boldsymbol{\delta}) - \mathbf{p}^*\|_2 + \mathbf{1}_{r > \epsilon_x} K, \quad \forall \|\boldsymbol{\delta}\|_2 \leq r,$$

with $K = 150cm$, and $\alpha = 0.35$. For learning of $\underline{p_A}$ using Clopper-Pearson ($\beta = 0.5$: 75% confidence), we used 100 samples and then we used $n = 10$ per radius to examine $\ell_2$ attack. For each scene, the adopted parameters are selected differently to cover various experimental setups.

*Great Court:* $P = 0.8$, $\epsilon_y = 5m$, output $\ell_1$ norm, $\sigma = 0.05$, and 760 images sized $480 \times 854$.

*King's College:* $P = 0.8$, $\epsilon_y = 1m$, output $\ell_1$ norm, $\sigma = 0.08$, and 343 images sized $480 \times 854$.

*St. Mary Church:* $P = 0.9$, $\epsilon_y = 5m$, output $\ell_1$ norm, $\sigma = 0.1$, and 530 images sized $480 \times 854$.

Figure 5 illustrates the proposed certified radii along the predicted trajectory of the camera in the 3 considered scenes as well as sample images with no/negligible certificates. From the top: Great Court, King's College, and St. Mary Church. While brighter dots represent points with higher certificates, we did not clip the radii from below and kept the negative values as they are. Large negative radii are indicators of high expectations of users either in considered valid regions or the expected probability of success. While most of the points are highlighted with bright colors, some points are dark and considered as sensitive images which can be easily misled with a small portion of manipulation. Interestingly, all the points which are not on the trajectory of the camera, have returned negative radii. On the other hand, Figure 6 illustrates the certified median error of the proposed methods for the three scenes in comparison with the results shown in [17] for bounded and discounted outputs for the Great Court scene (discount factor makes the accepted region wider than that of the base regression model, aiming for better analytical results in worst-case scenarios). Note that results in RS-Reg [17] were obtained with 200% discount in the output validity range, and we now provide almost the same results with no discount in the output, and results are valid for a small number of samples. As shown in these plots, the $\alpha$-trimming filter consistently decreased the certified median error (orange curve) across all input perturbation ranges ($r$) compared to the results obtained by the base regression model

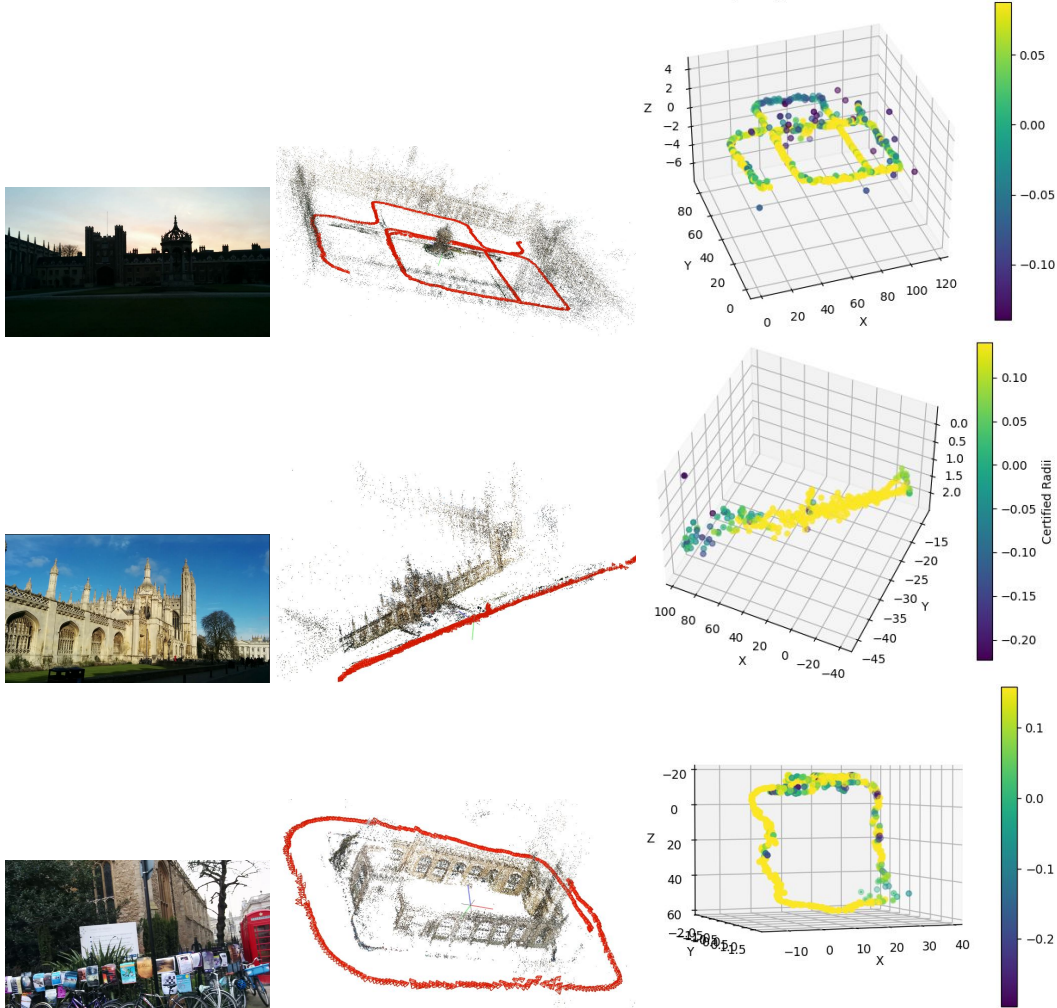

Figure 5: Evaluated scenes from the Cambridge Landmarks dataset (from the top row: Great Court, King's College, and St. Mary Church). The middle column depicts the reconstructed 3d sparse point clouds using Structure-from-Motion (SfM) where the images are taken in red trajectories (predicted). The right column visualizes the derived certified radii for each image taken on these trajectories. For some images, they have shown robustness to input perturbation (bright points), and for some images, they have shown sensitive results (dark points). Examples of images with no/negligible certificates are provided in the left column. As shown, these images suffer from lighting conditions, improper perspective, and obstructed content.

(blue curve). The main reasons for this improvement are firstly, due to the better approximation of position parameters leveraging outlier removal and averaging using $\alpha$-trimming filter. Secondly, because of better certificate radii for each image in the scene which decreases penalization in the process of certified median error calculation. Leveraging the $\alpha$-trimming approach for smoothing, we are no longer worried about the output ranges, and no further assumptions such as large sample size or discount factor are required to provide a valid certificate. Sensitivity analysis of the proposed technique in this dataset can be found in Appendix E.

## 5 Related Work

Among the studies that adopted randomized smoothing for tasks other than classification, we can list smoothed embeddings [20] for few-shot learning models, certification of soft classifiers [25] where the output variables are continuous but bounded between 0 and 1, certification against poisoning

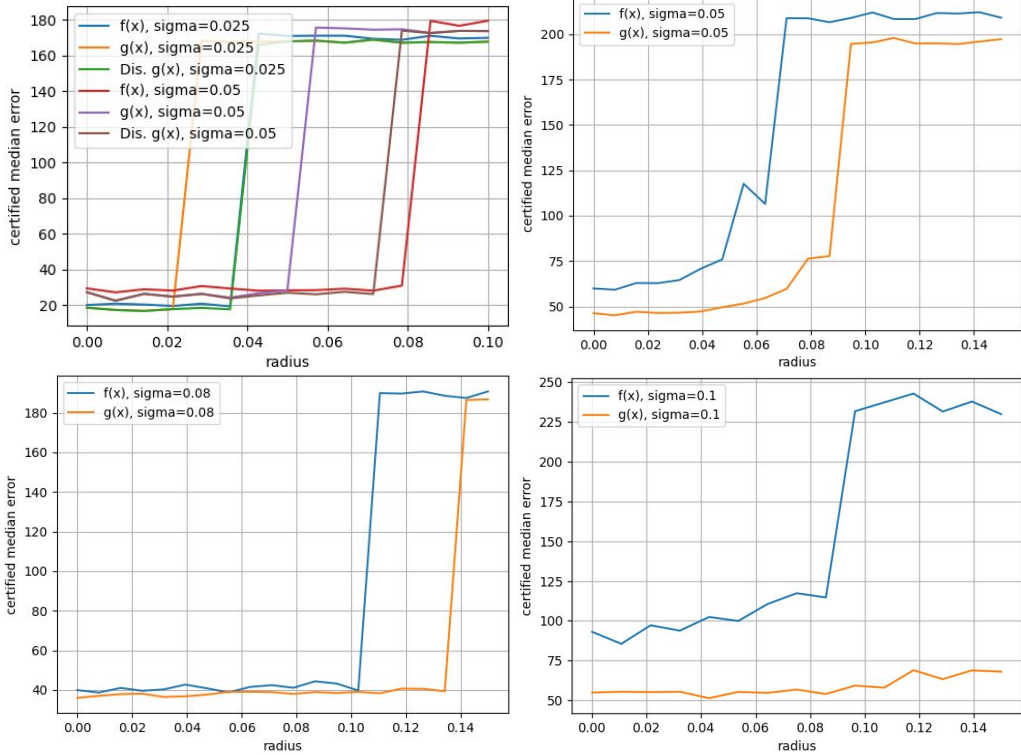

Figure 6: Certified median error in DSAC* as a function of $r$. These plots are for (top-left) RS-Reg with $200\%$ discount in Great Court scene (figure from [17]), (top-right) proposed method in Great Court scene, (bottom-left) proposed method in King's College scene, and (bottom-right) proposed method in St. Mary Church scene. The parameters are $\alpha = 0.35$, $n = 10$, $K = 150cm$, and $\beta = 0.5$.

attacks [7] which have different threat models or learning tasks than this study. The most related works to our study are [4, 17] where in the former study, the object detection was investigated through the lens of certified regression, however, their analysis is relying on the scaling output of classifier models to expand the range of output values which constrains the architecture of considered models, and in the latter the certification was provided for a class of bounded output regression model in the asymptotic case. Compared to these methods, our approach provides a probabilistic certificate for all regression models (including models with a wide range of outputs) with a limited number of evaluations through drawing noisy samples.

## 6 Conclusion

In this paper, we proposed the first probabilistic certificate against $\ell_p$ attack for all regression models with continuous output. We showed that the $\alpha$-trimming filter is an appropriate smoothing candidate in regression models with the flexibility to trade-off between error rate and certification radii. In addition to the comprehensive synthetic simulations, we adopted the proposed method in the camera re-localization task using the Cambridge Landmarks dataset and benchmarked the result for this new line of research. As future work, this technique can be extended to attacks in the semantic space of the input, and for sampling techniques other than Gaussian sampling to further tighten the certificate radii.

**Broader Impact** Adversarial examples demonstrate the vulnerability of many machine learning models to manipulation in contested environments. This paper considers defenses (via randomized smoothing) and robustness quantification (via robustness certificates), which are important approaches to improving resistance to attacks, and in highlighting the limitations of learned models to practitioners. As such, we believe this work has potential for positive societal benefit.

## Acknowledgement

This work was supported in part by the Department of Industry, Science, and Resources, Australia under AUSMURI CATCH, and the Australian Research Council under Discovery Project DP220102269.

## References

[1] Bednar, J., Watt, T.: Alpha-trimmed means and their relationship to median filters. IEEE Transactions on Acoustics, Speech, and Signal Processing **32**(1), 145–153 (1984)

[2] Biggio, B., Corona, I., Maiorca, D., Nelson, B., Šrndić, N., Laskov, P., Giacinto, G., Roli, F.: Evasion attacks against machine learning at test time. In: European Conference on Machine Learning and Knowledge Discovery in Databases. pp. 387–402 (2013)

[3] Brachmann, E., Rother, C.: Visual camera re-localization from RGB and RGB-D images using DSAC. IEEE Transactions on Pattern Analysis and Machine Intelligence **44**(9), 5847–5865 (2022)

[4] Chiang, P.y., Curry, M., Abdelkader, A., Kumar, A., Dickerson, J., Goldstein, T.: Detection as regression: Certified object detection with median smoothing. Advances in Neural Information Processing Systems **33**, 1275–1286 (2020)

[5] Cinà, A.E., Grosse, K., Demontis, A., Vascon, S., Zellinger, W., Moser, B.A., Oprea, A., Biggio, B., Pelillo, M., Roli, F.: Wild patterns reloaded: A survey of machine learning security against training data poisoning. ACM Computing Surveys **55**(13s), 1–39 (2023)

[6] Cohen, J., Rosenfeld, E., Kolter, Z.: Certified adversarial robustness via randomized smoothing. In: International Conference on Machine Learning. pp. 1310–1320. PMLR (2019)

[7] Hammoudeh, Z., Lowd, D.: Reducing certified regression to certified classification for general poisoning attacks. In: 2023 IEEE Conference on Secure and Trustworthy Machine Learning (SaTML). pp. 484–523. IEEE (2023)

[8] Huang, Z., Marchant, N.G., Lucas, K., Bauer, L., Ohrimenko, O., Rubinstein, B.I.: RS-Del: Edit distance robustness certificates for sequence classifiers via randomized deletion. In: Thirty-seventh Conference on Neural Information Processing Systems (2023)

[9] Kendall, A., Grimes, M., Cipolla, R.: PoseNet: A convolutional network for real-time 6-DOF camera relocalization. In: Proceedings of the IEEE International Conference on Computer Vision. pp. 2938–2946 (2015)

[10] Krizhevsky, A., Sutskever, I., Hinton, G.E.: ImageNet classification with deep convolutional neural networks. In: Advances in Neural Information Processing Systems (2012)

[11] Kumar, A., Levine, A., Goldstein, T., Feizi, S.: Curse of dimensionality on randomized smoothing for certifiable robustness. In: International Conference on Machine Learning. pp. 5458–5467. PMLR (2020)

[12] Lecuyer, M., Atlidakis, V., Geambasu, R., Hsu, D., Jana, S.: Certified robustness to adversarial examples with differential privacy. In: 2019 IEEE Symposium on Security and Privacy (SP). pp. 656–672. IEEE (2019)

[13] Lee, G.H., Yuan, Y., Chang, S., Jaakkola, T.: Tight certificates of adversarial robustness for randomly smoothed classifiers. Advances in Neural Information Processing Systems **32** (2019)

[14] Li, B., Chen, C., Wang, W., Carin, L.: Second-order adversarial attack and certifiable robustness (2018)

[15] Li, L., Xie, T., Li, B.: SoK: Certified robustness for deep neural networks. In: 2023 IEEE Symposium on Security and Privacy (SP). pp. 1289–1310. IEEE (2023)

[16] Liu, J., Levine, A., Lau, C.P., Chellappa, R., Feizi, S.: Segment and complete: Defending object detectors against adversarial patch attacks with robust patch detection. In: Proceedings of the IEEE/CVF Conference on Computer Vision and Pattern Recognition. pp. 14973–14982 (2022)

[17] Miri Rekavandi, A., Ohrimenko, O., Rubinstein, B.I.: RS-Reg: Probabilistic and robust certified regression through randomized smoothing. arXiv preprint arXiv:2405.08892 (2024)

[18] Mohapatra, J., Ko, C.Y., Weng, T.W., Chen, P.Y., Liu, S., Daniel, L.: Higher-order certification for randomized smoothing. Advances in Neural Information Processing Systems **33**, 4501–4511 (2020)

[19] Nadeem, U., Bennamoun, M., Togneri, R., Sohel, F., Miri Rekavandi, A., Boussaid, F.: Cross domain 2D-3D descriptor matching for unconstrained 6-DOF pose estimation. Pattern Recognition **142**, 109655 (2023)

[20] Pautov, M., Kuznetsova, O., Tursynbek, N., Petiushko, A., Oseledets, I.: Smoothed embeddings for certified few-shot learning. Advances in Neural Information Processing Systems **35**, 24367–24379 (2022)

[21] Rekavandi, A.M., Boussaid, F., Seghouane, A.K., Bennamoun, M.: B-Pose: Bayesian deep network for camera 6-DoF pose estimation from RGB images. IEEE Robotics and Automation Letters (2023)

[22] Rekavandi, A.M., Seghouane, A.K., Abed-Meraim, K.: TRPAST: A tunable and robust projection approximation subspace tracking method. IEEE Transactions on Signal Processing **71**, 2407–2419 (2023)

[23] Rekavandi, A.M., Seghouane, A.K., Evans, R.J.: Robust subspace detectors based on $\alpha$-divergence with application to detection in imaging. IEEE Transactions on Image Processing **30**, 5017–5031 (2021)

[24] Rekavandi, A.M., Seghouane, A.K., Evans, R.J.: Learning robust and sparse principal components with the $\alpha$-divergence. IEEE Transactions on Image Processing (2024)

[25] Salman, H., Li, J., Razenshteyn, I., Zhang, P., Zhang, H., Bubeck, S., Yang, G.: Provably robust deep learning via adversarially trained smoothed classifiers. Advances in Neural Information Processing Systems **32** (2019)

[26] Szegedy, C., Zaremba, W., Sutskever, I., Bruna, J., Erhan, D., Goodfellow, I., Fergus, R.: Intriguing properties of neural networks. In: International Conference on Learning Representations (2014)

[27] Tanielian, U., Biau, G.: Approximating Lipschitz continuous functions with GroupSort neural networks. In: International Conference on Artificial Intelligence and Statistics. pp. 442–450. PMLR (2021)

[28] Teng, J., Lee, G.H., Yuan, Y.: $\ell_1$ adversarial robustness certificates: a randomized smoothing approach (2019)

[29] Wheeden, R.L., Zygmund, A.: Measure and integral, vol. 26. Dekker New York (1977)

[30] Yang, G., Duan, T., Hu, J.E., Salman, H., Razenshteyn, I., Li, J.: Randomized smoothing of all shapes and sizes. In: International Conference on Machine Learning. pp. 10693–10705. PMLR (2020)

[31] Zhang, D., Ye, M., Gong, C., Zhu, Z., Liu, Q.: Black-box certification with randomized smoothing: A functional optimization based framework. Advances in Neural Information Processing Systems **33**, 2316–2326 (2020)

[32] Zoubir, A.M., Koivunen, V., Chakhchoukh, Y., Muma, M.: Robust estimation in signal processing: A tutorial-style treatment of fundamental concepts. IEEE Signal Processing Magazine **29**(4), 61–80 (2012)

# A    Proof of Robustness in $\mathbf{f}_\theta(\mathbf{x})$ against $\ell_p$ attack

We repeat the proposition's statement here for convenience, followed by its proof.

**Proposition 1:** *(Certification of $\boldsymbol{f}_\theta(\boldsymbol{x})$ Against $\ell_p$ Attack). Let $\boldsymbol{f}_\theta(\boldsymbol{x}) : \mathbb{R}^d \to \mathbb{R}^t$ be a (possibly) randomized base regressor and $\boldsymbol{e} \sim \mathcal{N}(\boldsymbol{0}, \sigma^2 \boldsymbol{I})$. Suppose*

$$\mathbb{P}\{diss_y(\boldsymbol{f}_\theta(\boldsymbol{x}+\boldsymbol{e})_i, \boldsymbol{y}_i) \leq \epsilon_{y_i}\} \geq \underline{p_{A_i}}, \forall i \in [\![t]\!], \tag{15}$$

*where $\underline{p_{A_i}}$ is the lower bound on the probability of accepting prediction in $i^{th}$ output variable. Then using (2) as the definition of certified robustness, $\boldsymbol{f}_\theta(\boldsymbol{x}+\boldsymbol{\delta}+\boldsymbol{e})$, $\forall \|\boldsymbol{\delta}\|_p \leq \epsilon_x$ ($p \geq 2$) is within the accepted region, with the user-defined probability $P \leq \underline{p_{A_i}}, \forall i \in [\![t]\!]$, where*

$$\epsilon_x = \min_{i \in [\![t]\!]} \frac{\sigma}{d^{\frac{1}{2}-\frac{1}{p}}} \left( \Phi^{-1}(\underline{p_{A_i}}) - \Phi^{-1}(P) \right). \tag{16}$$

*Proof.* For any $\mathbf{x}, \mathbf{y} \in \mathbb{R}^d$, let us define $\mathbf{x}*\mathbf{y} = (x_i y_i)_{i=1,\cdots,d} \in \mathbb{R}^d$. Then using the generalization of Hoelder inequality [29], for any $p, q, r \in [1, \infty)$ such that $\frac{1}{p} + \frac{1}{q} = \frac{1}{r}$, we have $\|\mathbf{x}*\mathbf{y}\|_r \leq \|\mathbf{x}\|_p \|\mathbf{y}\|_q$. By setting $\mathbf{x} = \boldsymbol{\delta}$ and $\mathbf{y} = \mathbf{1}_d$, we obtain $\|\boldsymbol{\delta}\|_r \leq d^{\frac{1}{r}-\frac{1}{p}} \|\boldsymbol{\delta}\|_p$, (with $\frac{1}{q} = \frac{1}{r} - \frac{1}{p}$). Using this inequality, and by setting $r = 2$, the constraint ($p \geq 2$), and the results in Theorem 1, we conclude that if $d^{\frac{1}{2}-\frac{1}{p}} \|\boldsymbol{\delta}\|_p \leq \min_{i \in [\![t]\!]} \sigma \left( \Phi^{-1}(\underline{p_{A_i}}) - \Phi^{-1}(P) \right)$, the output value is valid with probability at least P. This completes the proof. $\qquad\square$

# B    Proof of Robustness for $\mathbf{g}_\alpha(\mathbf{x})$

We repeat the theorem's statement here for convenience, followed by its proof.

**Theorem 2:** (Certification of $\mathbf{g}_\alpha(\mathbf{x})$ for fixed range $\ell_p$ Attack). *Let $\boldsymbol{f}_\theta(\boldsymbol{x}) : \mathbb{R}^d \to \mathbb{R}^t$ be a (possibly) randomized base regressor, and let $\boldsymbol{g}_\alpha(\boldsymbol{x})$ be the one defined in (8) with $0 \leq \alpha \leq 1/2$. Let us assume the accepted region (set) for each output target variable is convex and the following inequality holds for $\boldsymbol{e} \sim \mathcal{N}(\boldsymbol{0}, \sigma^2 \boldsymbol{I})$, a user-defined $\boldsymbol{\epsilon}_y$ and $\forall \|\boldsymbol{\delta}\|_p \leq \epsilon_x$ ($p \geq 2$) and $\forall i \in [\![t]\!]$:*

$$\mathbb{P}\{diss_y(\boldsymbol{f}_\theta(\boldsymbol{x}+\boldsymbol{\delta}+\boldsymbol{e})_i, \boldsymbol{y}_i) \leq \boldsymbol{\epsilon}_{y_i}\} \geq q, \tag{17}$$

*where $0 \leq q \leq 1$, then $\forall n > 0$ we have*

$$\mathbb{P}\{diss_y(\mathbf{g}_\alpha(\mathbf{x}+\boldsymbol{\delta})_i, \mathbf{y}_i) \leq \boldsymbol{\epsilon}_{y_i}\} \geq I_q(n - [\alpha n], [\alpha n] + 1), \forall i \in [\![t]\!] \tag{18}$$

*where $I_q(a,b)$ is the regularized incomplete beta function defined as $I_q(a,b) = \frac{1}{B(a,b)} \int_0^q t^{a-1}(1-t)^{b-1}dt$ and $B(a,b)$ is the complete beta function.*

*Proof.* The proof of this theorem is based on the problem's geometry and admitting that simple averaging is prone to be shifted outside of the accepted set, only if a single large sample exists in the remaining samples after applying $\alpha$-trimming. To prove the result, we consider the worst-case scenario and derive the lower bound probability of getting an accepted output within the desired range defined by users. Let us define a random variable $W$ as the number of rejected outputs by the user (invalid outputs) after drawing $n$ samples and passing then through the network $\mathbf{f}_\theta(.)$. For a scalar output variable, three possible scenarios will occur after sorting output values, (i) all rejected outputs will be on the left side of the accepted region, (ii) all rejected outputs will be on the right side of the accepted region, (iii) rejected outputs will be both on the left and right with different portions. Figure 7 illustrates the geometry of these scenarios. Note that since the accepted zone is convex (green region), all the accepted outputs will be next to each other after sorting. As shown in (8), the trimming will be applied equally from both sides, so in the worst-case scenario (scenarios (i) and (ii)) both accepted and rejected outputs will be filtered with the same rate. In this case, in total $2[\alpha n]$ outputs will be filtered out ($[\alpha n]$ accepted outputs and $[\alpha n]$ rejected outputs) where the total number of leftover samples will be $n - 2[\alpha n]$. Now let us consider two cases:
 1. The case $W > [\alpha n]$: In this case after applying $\alpha$-trimming, still $W - [\alpha n]$ rejected samples will remain in the bag (set of leftover data points after applying $\alpha$-trimming filter) to be averaged, and

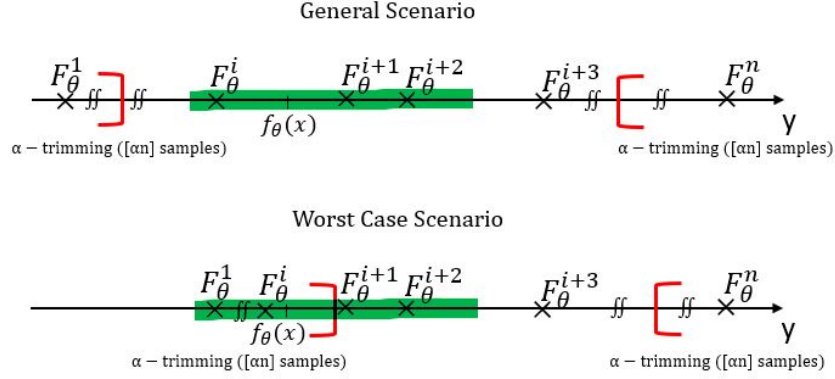

Figure 7: Potential scattering plots for sorted outputs of a given regression model when fed with perturbed inputs. The general scenario which is most likely the case (top), and the worst-case scenario (bottom).

since in the worst-case scenario one single sample is enough to push the average value outside the accepted region, for this case we consider

$$\mathbb{P}\{\mathrm{diss}_y(\mathbf{g}_\alpha(\mathbf{x}+\boldsymbol{\delta})_i, \mathbf{y}_i) \leq \epsilon_{y_i} \mid W > [\alpha n]\} = 0. \tag{19}$$

2. The case $W \leq [\alpha n]$: In this case, there will be no rejected samples left in the bag to push the average outside the acceptable zone. Since all the output values are already within the convex set, the average value is also within the set and the average will always be valid to the user. Therefore, in this case,

$$\mathbb{P}\{\mathrm{diss}_y(\mathbf{g}_\alpha(\mathbf{x}+\boldsymbol{\delta})_i, \mathbf{y}_i) \leq \epsilon_{y_i} \mid W \leq [\alpha n]\} = 1. \tag{20}$$

The only variable that can put us in either of these two disjoint scenarios is the random variable $W$ ($n$ and $\alpha$ are predefined values), which follows a Binomial distribution and in worst case scenario the success rate is $q$ as assumed in Theorems's statement, i.e., $W \sim \mathrm{Bin}(n, 1-q)$. Hence, $\forall i \in [\![t]\!]$ we have

$$
\begin{aligned}
&\mathbb{P}\{\mathrm{diss}_y(\mathbf{g}_\alpha(\mathbf{x}+\boldsymbol{\delta})_i, \mathbf{y}_i) \leq \epsilon_{y_i}\} \\
=\ &\sum_{k=0}^{n} \mathbb{P}\{\mathrm{diss}_y(\mathbf{g}_\alpha(\mathbf{x}+\boldsymbol{\delta})_i, \mathbf{y}_i) \leq \epsilon_{y_i} \mid W=k\}\mathbb{P}\{W=k\} \\
\geq\ &\sum_{k=0}^{n} \mathbb{P}\{\mathrm{diss}_y(\mathbf{g}_\alpha(\mathbf{x}+\boldsymbol{\delta})_i, \mathbf{y}_i) \leq \epsilon_{y_i} \mid W=k\}\binom{n}{k}(1-q)^k q^{n-k} \\
=\ &\sum_{k=0}^{k\leq[\alpha n]} \mathbb{P}\{\mathrm{diss}_y(\mathbf{g}_\alpha(\mathbf{x}+\boldsymbol{\delta})_i, \mathbf{y}_i) \leq \epsilon_{y_i} \mid W=k\}\binom{n}{k}(1-q)^k q^{n-k} \\
&+ \sum_{k>[\alpha n]}^{n} \mathbb{P}\{\mathrm{diss}_y(\mathbf{g}_\alpha(\mathbf{x}+\boldsymbol{\delta})_i, \mathbf{y}_i) \leq \epsilon_{y_i} \mid W=k\}\binom{n}{k}(1-q)^k q^{n-k} \\
\geq\ &\sum_{k=0}^{[\alpha n]} \binom{n}{k}(1-q)^k q^{n-k} \\
=\ &I_q(n-[\alpha n], [\alpha n]+1), \tag{21}
\end{aligned}
$$

where $I_q(a,b)$ is the regularized incomplete beta function defined as $I_q(a,b) = \frac{1}{B(a,b)}\int_0^q t^{a-1}(1-t)^{b-1}dt$ to compute the cumulative distribution of a binomial distribution where $B(a,b)$ is the complete beta function.

$\square$

## C Robustness Improvement via $\alpha$-trimming

We repeat the proposition's statement here for convenience, followed by its proof.

**Proposition 2:** (Robustness Improvement via $\alpha$-trimming). *For any $n \geq 1$, $0 \leq q \leq 1$, if there exist an $\alpha$ such that $\alpha^+ \leq \alpha < 1/2$, where*

$$\alpha^+ := \frac{I_q^{-1}(q) - 1/2}{n}, \tag{22}$$

*and where $I_q^{-1}(q)$ is the inverse of $I_q(n - x, x + 1)$ with respect to $x$, then the following inequality holds:*

$$I_q(n - [\alpha n], [\alpha n] + 1) \geq q. \tag{23}$$

*Proof.* Note that $I_q(n - x, x + 1)$ where $x \in \mathbb{Z}^+$, is the CDF of $W \sim \text{Bin}(n, 1 - q)$, and is a monotonically increasing function with respect to $x$. To guarantee that the outcome CDF is greater than $q$, we first need to find the corresponding $x$ which achieves equality, and then find the region where the inequality is always correct. To find the smallest $x$, we solve

$$I_q(n - x, x + 1) = q \Rightarrow x = I_q^{-1}(q) \tag{24}$$

Knowing that in our formulation $x$ corresponds to $[\alpha n]$, then to meet the Equality (24), $\alpha$ is required to meet

$$I_q^{-1}(q) - 1/2 \leq \alpha n < I_q^{-1}(q) + 1/2$$
$$\Rightarrow \quad \frac{I_q^{-1}(q) - 1/2}{n} \leq \alpha < \frac{I_q^{-1}(q) + 1/2}{n}. \tag{25}$$

For a fixed $n$ and $q$, this means the inequality $I_q(n - [\alpha n], [\alpha n] + 1) \geq q$ holds if we select an $\alpha$ such that it satisfies $\alpha \geq \frac{I_q^{-1}(q) - 1/2}{n}$ and this completes the proof. $\square$

## D Certification of $\mathbf{g}_\alpha(\mathbf{x})$

We repeat the theorem's statement here for convenience, followed by its proof.

**Theorem 3:** (Certification of $\mathbf{g}_\alpha(\mathbf{x})$ against $\ell_p$ Attack). *Let $f_\theta(\boldsymbol{x}) : \mathbb{R}^d \to \mathbb{R}^t$ be a deterministic or random base regressor and let $n \geq 1$, $0 \leq \alpha < 1/2$, $\boldsymbol{e} \sim \mathcal{N}(\boldsymbol{0}, \sigma^2 \boldsymbol{I})$ and suppose the $\alpha$-trimming function $\boldsymbol{g}_\alpha(\boldsymbol{x})$ defined in (8) is used for smoothing. Then given*

$$\mathbb{P}\{diss_y(\boldsymbol{f}_\theta(\boldsymbol{x} + \boldsymbol{e})_i, \boldsymbol{y}_i) \leq \epsilon_{y_i}\} \geq \underline{p_{A_i}}, \forall i \in [\![t]\!] \tag{26}$$

*where $\underline{p_{A_i}}$ is the lower bound on the probability of accepting prediction in the $i^{th}$ output variable, then $\boldsymbol{g}_\alpha(\boldsymbol{x} + \boldsymbol{\delta})$, $\forall \|\boldsymbol{\delta}\|_p \leq \epsilon_x$ ($p \geq 2$) is within accepted region, i.e., $\mathbf{N}_y(\boldsymbol{y}, \epsilon_y) = \prod_{i=1}^t \mathbf{N}_y(\boldsymbol{y}_i, \epsilon_{y_i})$, with the user-defined probability P, s.t. $I_{n,\alpha}^{-1}(P) \leq \underline{p_{A_i}}, \forall i \in [\![t]\!]$, where*

$$\epsilon_x = \min_{i \in [\![t]\!]} \frac{\sigma}{d^{\frac{1}{2} - \frac{1}{p}}} \left( \Phi^{-1}(\underline{p_{A_i}}) - \Phi^{-1}(I_{n,\alpha}^{-1}(P)) \right), \tag{27}$$

*and where $I_{n,\alpha}^{-1}(x)$ is the inverse of the regularized beta function w.r.t Bernoulli success rate parameter.*

*Proof.* We consider the following chain to derive this upper bound on the $\ell_p$ norm of $\boldsymbol{\delta}$.

$$\mathbf{x} + \boldsymbol{\delta}, \|\boldsymbol{\delta}\|_p < \epsilon_x \xrightarrow{\mathbf{f}_\theta(\mathbf{x})} \mathbb{P}\{\text{Valid } \mathbf{f}_\theta(\mathbf{x})\} \geq q \xrightarrow{\mathbf{g}_\alpha(\mathbf{x})} \mathbb{P}\{\text{Valid } \mathbf{g}_\alpha(\mathbf{x})\} \geq I_q(n - [\alpha n], [\alpha n] + 1). \tag{28}$$

Based on the Proposition 1, if we select $\|\boldsymbol{\delta}\|_p \leq \min_{i \in [\![t]\!]} \frac{\sigma}{d^{\frac{1}{2} - \frac{1}{p}}} \left( \Phi^{-1}(\underline{p_{A_i}}) - \Phi^{-1}(q) \right)$, then $\mathbf{f}_\theta(\mathbf{x})$ is valid with probability $q$, and then based on results in Theorem 2, $\mathbf{g}_\alpha(\mathbf{x})$ changes this probability

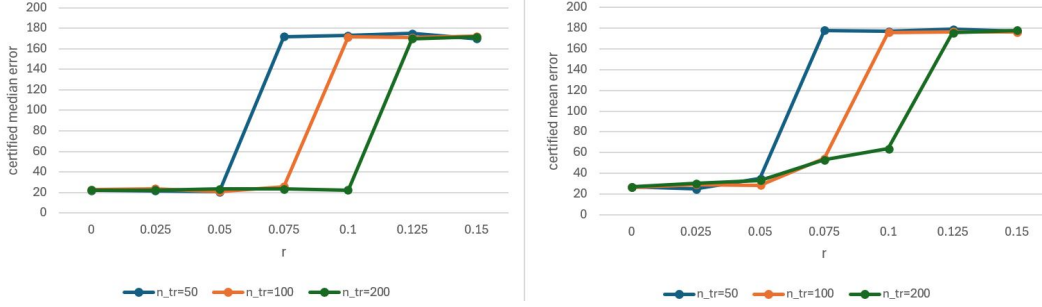

Figure 8: Sensitivity of certified median (left) and mean (right) error with changes in $n_{tr}$.

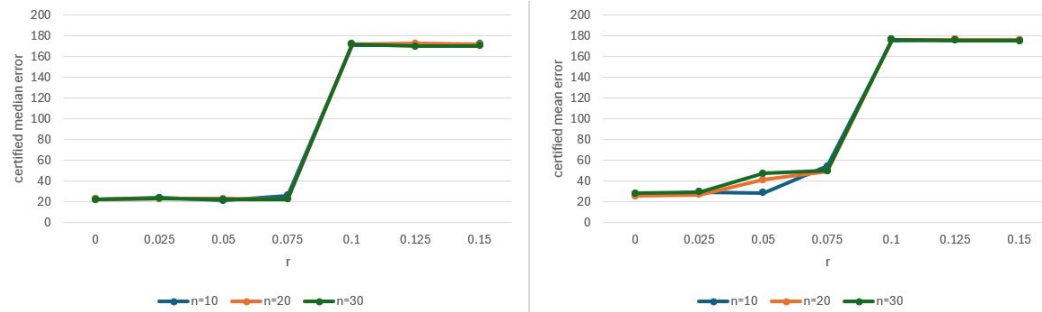

Figure 9: Sensitivity of certified median (left) and mean (right) error with changes in $n$.

into $I_q(n - [\alpha n], [\alpha n] + 1)$ for a fixed $\alpha$ and $n$, Therefore, if one asks the $\mathbf{g}_\alpha(\mathbf{x})$ to be valid with probability $P$, we first solve

$$I_q(n - [\alpha n], [\alpha n] + 1) = P, \tag{29}$$

for $0 \le q \le 1$. Note that $I_q(n - [\alpha n], [\alpha n] + 1) \propto \int_0^q t^{n-[\alpha n]-1}(1 - t)^{[\alpha n]}dt$ and the right hand side is monotonically increasing function w.r.t $q$ when $n$ and $\alpha$ are fixed. Therefore, we can define the inverse $I_{n,\alpha}^{-1}(x) = q$ such that $I_q(n - [\alpha n], [\alpha n] + 1) = x$. Then, knowing this inverse, we can directly use Proposition 1 to find the bound on $\|\boldsymbol{\delta}\|_p$, and this completes the proof. $\qquad\square$

# E  Sensitivity Analysis

In this section, we perform a sensitivity analysis of the proposed robust certification technique for the hyperparameters involved in our mechanism. The parameters under evaluation are number of samples to estimate $p_A$ ($n_{tr}$), number of samples for $\alpha$-trimming filter ($n$), rate of filtering ($\alpha$), and confidence level parameter ($\beta$). We use the Great Court scene with 50 randomly selected images for examination with the size of $480 \times 854$. The default parameters are $K = 150cm$, $\alpha = 0.35$, $\beta = 0.5$, $n_{tr} = 100$, $n = 10$, $P = 0.8$, $\epsilon_y = 5m$ using $\ell_1$ norm in output, $\sigma = 0.05$, with robustness against $\ell_2$ attack. For each parameter, we only change the parameter of interest and keep the other fixed.

$n_{tr}$: This is the parameter that determines the quality of the estimated $P_{A_i}$. In the reported empirical results, this parameter was set to 100. Here we examine the performance for values in the set $\{50, 100, 200\}$ and report the certified median/mean error curve. Generally, we expect by increasing $n_{tr}$ no changes occur in the tails of the certified mean/median since this only determines how reliable the observed number of valid outputs is. If $n_{tr}$ is small, to keep the confidence level fixed, the estimated lower bound of $p_A$ will be sacrificed, and that means the estimated radii will be small. While this does not affect the curves' tails, the middle-range errors will be significantly increased due to the penalty term imposed by the range of radii. Figure 8 validates this interpretation and depicts this effect for the middle parts of the curves.

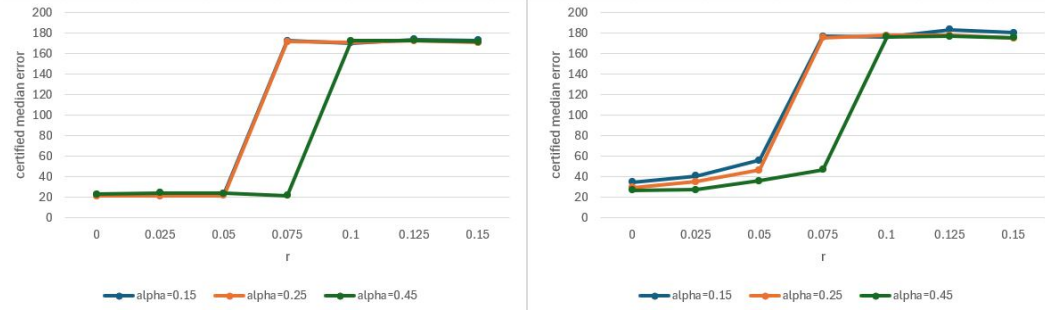

Figure 10: Sensitivity of certified median (left) and mean (right) error with changes in $\alpha$.

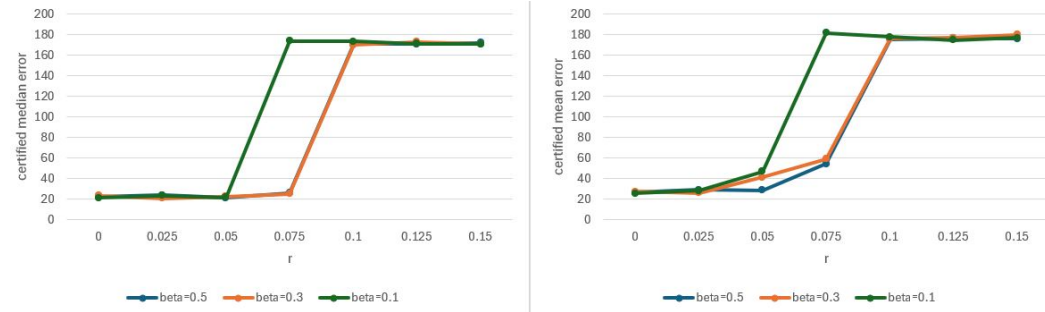

Figure 11: Sensitivity of certified median (left) and mean (right) error with changes in $\beta$.

$n$: This parameter indicates the number of samples which is going to be drawn in the output of the base regression model to be fed into the $\alpha$-trimming filter. In the experimental results, this was set to $n = 10$. Here we investigate the impact of this parameter in the obtained certified median/mean error rate by choosing the values among the set $\{10, 20, 30\}$. As shown in Figure 9 the obtained results are quite stable with changes in $n$ for the considered range.

$\boldsymbol{\alpha}$: Parameter $\alpha$ in $\alpha$-trimming filter determines the rate of sample rejection on both sides of the sorted values. While in the experiments, this value was set to $\alpha = 0.35$, in this analysis we investigate the role of this parameter in the final error rate results by choosing the values from the $\{0.15, 0.25, 0.45\}$. As illustrated in Figure 10, in terms of median error, for smaller $\alpha$ when $r$ is also small, we obtained a slightly better error rate because of having more samples in the average computation of the filtered values. On the other hand, increasing $\alpha$ leads to better certification radii and the jump to higher error rates occurred in a larger range of $r$ values. In terms of mean error rate, increasing the value of $\alpha$ uniformly improved the performance because all the outliers came out of the bag of averaging.

$\boldsymbol{\beta}$: This parameter indicates the amount of confidence required about the obtained results as compensation for the limited number of samples used in estimating $p_{A_i}$. In the experiments, this value was set to $\beta = 0.5$ to provide confidence of $1 - \frac{\beta}{2} = 0.75$. Now we choose this value from the set $\{0.1, 0.3, 0.5\}$ to examine the robustness of the results. We expect as we decrease $\beta$, as the result of increasing the required confidence level, the estimated lower bound of $p_{A_i}, \forall i \in [\![t]\!]$ decreases and the certified radii decrease as a result. These smaller certified radii cause more rapid growth in the certified error rate as depicted in Figure 11.

